# Improvements to the Sequence Memoizer

**Jan Gasthaus**
Gatsby Computational Neuroscience Unit
University College London
London, WC1N 3AR, UK
j.gasthaus@gatsby.ucl.ac.uk

**Yee Whye Teh**
Gatsby Computational Neuroscience Unit
University College London
London, WC1N 3AR, UK
ywteh@gatsby.ucl.ac.uk

## Abstract

The sequence memoizer is a model for sequence data with state-of-the-art performance on language modeling and compression. We propose a number of improvements to the model and inference algorithm, including an enlarged range of hyperparameters, a memory-efficient representation, and inference algorithms operating on the new representation. Our derivations are based on precise definitions of the various processes that will also allow us to provide an elementary proof of the "mysterious" coagulation and fragmentation properties used in the original paper on the sequence memoizer by Wood et al. (2009). We present some experimental results supporting our improvements.

## 1 Introduction

The sequence memoizer (SM) is a Bayesian nonparametric model for discrete sequence data producing state-of-the-art results for language modeling and compression [1, 2]. It models each symbol of a sequence using a predictive distribution that is conditioned on all previous symbols, and thus can be understood as a non-Markov sequence model. Given the very large (infinite) number of predictive distributions needed to model arbitrary sequences, it is essential that statistical strength be shared in their estimation. To do so, the SM uses a hierarchical Pitman-Yor process prior over the predictive distributions [3]. One innovation of the SM over [3] is its use of coagulation and fragmentation properties [4, 5] that allow for efficient representation of the model using a data structure whose size is linear in the sequence length. However, in order to make use of these properties, all concentration parameters, which were allowed to vary freely in [3], were fixed to zero.

In this paper we explore a number of further innovations to the SM. Firstly, we propose a more flexible setting of the hyperparameters with potentially non-zero concentration parameters that still allow the use of the coagulation/fragmentation properties. In addition to better predictive performance, the setting also partially mitigates a problem observed in [1], whereby on encountering a long sequence of the same symbol, the model becomes overly confident that it will continue with the same symbol.

The second innovation addresses memory usage issues in inference algorithms for the SM. In particular, current algorithms use a Chinese restaurant franchise representation for the HPYP, where the seating arrangement of customers in each restaurant is represented by a list, each entry being the number of customers sitting around one table [3]. This is already an improvement over the naïve Chinese restaurant franchise in [6] which stores pointers from customers to the tables they sit at, but can still lead to huge memory requirements when restaurants contain many tables. One approach to mitigate this problem has been explored in [7], which uses a representation that stores a histogram of table sizes instead of the table sizes themselves. Our proposal is to store even less, namely only the minimal statistics about each restaurant required to make predictions: the number of customers and the number of tables occupied by the customers. Inference algorithms will have to be adapted to this compact representation, and we describe and compare a number of these.

In Section 2 we will give precise definitions of Pitman-Yor processes and Chinese restaurant processes. These will be used to define the SM model in Section 3, and to derive the results about the extended hyperparameter setting in Section 4 and the memory-efficient representation in Section 5. As a side benefit we will also be able to give an elementary proof of the coagulation and fragmentation properties in Section 4, which was presented as a fait accompli in [1], while the general and rigorous treatment in the original papers [4, 5] is somewhat inaccessible to a wider audience.

## 2 Pitman-Yor Processes and Chinese Restaurant Processes

A Pitman-Yor process (PYP) is a particular distribution over distributions over some probability space $\Sigma$ [8, 9]. We denote by $\mathrm{PY}(\alpha, d, G_0)$ a PYP with concentration parameter $\alpha > -d$, discount parameter $d \in [0, 1)$, and base distribution $G_0$ over $\Sigma$. We can describe a Pitman-Yor process using its associated Chinese restaurant process (CRP). A Chinese restaurant has customers sitting around tables which serve dishes. If there are $c$ customers we index them with $[c] = \{1, \ldots, c\}$. We define a seating arrangement of the customers as a set of disjoint non-empty subsets partitioning $[c]$. Each subset is a table and consists of the customers sitting around it, e.g. $\{\{1, 3\}, \{2\}\}$ means customers 1 and 3 sit at one table and customer 2 sits at another by itself. Let $\mathcal{A}_c$ be the set of seating arrangements of $c$ customers, and $\mathcal{A}_{ct}$ those with exactly $t$ tables. The CRP describes a distribution over seating arrangements as follows: customer 1 sits at a table; for customer $c + 1$, if $A \in \mathcal{A}_c$ is the current seating arrangement, then she joins a table $a \in A$ with probability $\frac{|a|-d}{\alpha+c}$ and starts a new table with probability $\frac{\alpha+|A|d}{\alpha+c}$. We denote the resulting distribution over $\mathcal{A}_c$ as $\mathrm{CRP}_c(\alpha, d)$. Multiplying the conditional probabilities together,

$$P(A) = \frac{[\alpha + d]_d^{|A|-1}}{[\alpha + 1]_1^{c-1}} \prod_{a \in A} [1 - d]_1^{|a|-1} \quad \text{for each } A \in \mathcal{A}_c, \tag{1}$$

where $[y]_d^n = \prod_{i=0}^{n-1} y + id$ is Kramp's symbol. Note that the denominator is the normalization constant. Fixing the number of tables to be $t \le c$, the distribution, denoted as $\mathrm{CRP}_{ct}(d)$, becomes:

$$P(A) = \frac{\prod_{a \in A}[1 - d]_1^{|a|-1}}{S_d(c, t)} \quad \text{for each } A \in \mathcal{A}_{ct}, \tag{2}$$

where the normalization constant $S_d(c, t) = \sum_{A \in \mathcal{A}_{ct}} \prod_{a \in A}[1 - d]_1^{|a|-1}$ is a generalized Stirling number of type $(-1, -d, 0)$ [10]. These can be computed recursively [3] (see also Section 5). Note that conditioning on a fixed $t$ the seating arrangement will not depend on $\alpha$, only on $d$.

Suppose $G \sim \mathrm{PY}(\alpha, d, G_0)$ and $z_1, \ldots, z_c | G \overset{\text{iid}}{\sim} G$. The CRP describes the PYP in terms of its effect on $z_{1:c} = z_1, \ldots, z_c$. In particular, marginalizing out $G$, the distribution of $z_{1:c}$ can be described as follows: draw $A \sim \mathrm{CRP}_c(\alpha, d)$, on each table serve a dish which is an iid draw from $G_0$, finally let variable $z_i$ take on the value of the dish served at the table that customer $i$ sat at. Now suppose we wish to perform inference given observation of $z_{1:c}$. This is equivalent to conditioning on the dishes that each customer is served. Since customers at the same table are served the same dish, the different values among the $z_i$'s split the restaurant into multiple sections, with customers and tables in each section being served a distinct dish. There can be more than one table in each section since multiple tables can serve the same dish (if $G_0$ has atoms). If $s \in \Sigma$ is a dish, let $c_s$ be the number of $z_i$'s with value $s$ (number of customers served dish $s$), $t_s$ the number of tables, and $A_s \in \mathcal{A}_{c_s t_s}$ the seating arrangement of customers around the tables serving dish $s$ (we reindex the $c_s$ customers to be $[c_s]$). The joint distribution over seating arrangements and observations is then:[1]

$$P(\{c_s, t_s, A_s\}, z_{1:c}) = \left( \prod_{s \in \Sigma} G_0(s)^{t_s} \right) \left( \frac{[\alpha + d]_d^{t.-1}}{[\alpha + 1]_1^{c.-1}} \prod_{s \in \Sigma} \prod_{a \in A_s} [1 - d]_1^{|a|-1} \right), \tag{3}$$

where $t. = \sum_{s \in \Sigma} t_s$ and similarly for $c.$. We can marginalize out $\{A_s\}$ from (3) using (2):

$$P(\{c_s, t_s\}, z_{1:c}) = \left( \prod_{s \in \Sigma} G_0(s)^{t_s} \right) \left( \frac{[\alpha + d]_d^{t.-1}}{[\alpha + 1]_1^{c.-1}} \prod_{s \in \Sigma} S_d(c_s, t_s) \right). \tag{4}$$

Inference then amounts to computing the posterior of either $\{t_s, A_s\}$ or only $\{t_s\}$ given $z_{1:c}$ ($c_s$ are fixed) and can be achieved by Gibbs sampling or other means.

## 3 The Sequence Memoizer and its Chinese Restaurant Representation

In this section we review the sequence memoizer (SM) and its representation using Chinese restaurants [3, 11, 1, 2]. Let $\Sigma$ be the discrete set of symbols making up the sequences to be modeled, and let $\Sigma^*$ be the set of finite sequences of symbols from $\Sigma$. The SM models a sequence $x_{1:T} = x_1, x_2, \ldots, x_T \in \Sigma^*$ using a set of conditional distributions:

$$P(x_{1:T}) = \prod_{i=1}^{T} P(x_i | x_{1:i-1}) = \prod_{i=1}^{T} G_{x_{1:i-1}}(x_i), \tag{5}$$

where $G_{\mathbf{u}}(s)$ is the conditional probability of the symbol $s \in \Sigma$ occurring after a context $\mathbf{u} \in \Sigma^*$ (the sequence of symbols occurring before $s$). The parameters of the model consist of all the conditional distributions $\{G_{\mathbf{u}}\}_{\mathbf{u} \in \Sigma^*}$, and are given a hierarchical Pitman-Yor process (HPYP) prior:

$$\begin{aligned} G_{\varepsilon} &\sim \mathrm{PY}(\alpha_{\varepsilon}, d_{\varepsilon}, H) \\ G_{\mathbf{u}} | G_{\sigma(\mathbf{u})} &\sim \mathrm{PY}(\alpha_{\mathbf{u}}, d_{\mathbf{u}}, G_{\sigma(\mathbf{u})}) \qquad\qquad \text{for } \mathbf{u} \in \Sigma^* \backslash \{\varepsilon\}, \end{aligned} \tag{6}$$

where $\varepsilon$ is the empty sequence, $\sigma(\mathbf{u})$ is the sequence obtained by dropping the first symbol in $\mathbf{u}$, and $H$ is the overall base distribution over $\Sigma$ (we take $H$ to be uniform over a finite $\Sigma$). Note that we have generalized the model to allow each $G_{\mathbf{u}}$ to have its own concentration and discount parameters, whereas [1, 2] worked with $\alpha_{\mathbf{u}} = 0$ and $d_{\mathbf{u}} = d_{|\mathbf{u}|}$ (i.e. context length-dependent discounts).

As in previous works, the hierarchy over $\{G_{\mathbf{u}}\}$ is represented using a Chinese restaurant franchise [6]. Each $G_{\mathbf{u}}$ has a corresponding restaurant indexed by $\mathbf{u}$. Customers in the restaurant are draws from $G_{\mathbf{u}}$, tables are draws from its base distribution $G_{\sigma(\mathbf{u})}$, and dishes are the drawn values from $\Sigma$. For each $s \in \Sigma$ and $\mathbf{u} \in \Sigma^*$, let $c_{\mathbf{u}s}$ and $t_{\mathbf{u}s}$ be the numbers of customers and tables in restaurant $\mathbf{u}$ served dish $s$, and let $A_{\mathbf{u}s} \in \mathcal{A}_{c_{\mathbf{u}s} t_{\mathbf{u}s}}$ be their seating arrangement. Each observation of $x_i$ in context $x_{1:i-1}$ corresponds to a customer in restaurant $x_{1:i-1}$ who is served dish $x_i$, and each table in each restaurant $\mathbf{u}$, being a draw from the base distribution $G_{\sigma(\mathbf{u})}$, corresponds to a customer in the parent restaurant $\sigma(\mathbf{u})$. Thus, the numbers of customers and tables have to satisfy the constraints

$$c_{\mathbf{u}s} = c_{\mathbf{u}s}^x + \sum_{\mathbf{v}:\sigma(\mathbf{v})=\mathbf{u}} t_{\mathbf{v}s}, \tag{7}$$

where $c_{\mathbf{u}s}^x = 1$ if $s = x_i$ and $\mathbf{u} = x_{1:i-1}$ for some $i$, and 0 otherwise.

The goal of inference is to compute the posterior over the states $\{c_{\mathbf{u}s}, t_{\mathbf{u}s}, A_{\mathbf{u}s}\}_{s \in \Sigma, \mathbf{u} \in \Sigma^*}$ of the restaurants (and possibly the concentration and discount parameters). The joint distribution can be obtained by multiplying the probabilities of all seating arrangements (3) in all restaurants:

$$P(\{c_{\mathbf{u}s}, t_{\mathbf{u}s}, A_{\mathbf{u}s}\}, x_{1:T}) = \left( \prod_{s \in \Sigma} H(s)^{t_{\varepsilon s}} \right) \prod_{\mathbf{u} \in \Sigma^*} \left( \frac{[\alpha_{\mathbf{u}} + d_{\mathbf{u}}]_{d_{\mathbf{u}}}^{t_{\mathbf{u}\cdot}-1}}{[\alpha_{\mathbf{u}} + 1]_1^{c_{\mathbf{u}\cdot}-1}} \prod_{s \in \Sigma} \prod_{a \in A_{\mathbf{u}s}} [1 - d_{\mathbf{u}}]_1^{|a|-1} \right). \tag{8}$$

The first parentheses contain the probability of draws from the overall base distribution $H$, and the second parentheses contain the probability of the seating arrangement in restaurant $\mathbf{u}$. Given a state of the restaurants drawn from the posterior, the predictive probability of symbol $s$ in context $\mathbf{v}$ can then be computed recursively (with $P_{\sigma(\varepsilon)}^*(s)$ defined to be $H(s)$):

$$P_{\mathbf{v}}^*(s) = \frac{c_{\mathbf{v}s} - t_{\mathbf{v}s}d}{\alpha_{\mathbf{v}} + c_{\mathbf{v}\cdot}} + \frac{\alpha_{\mathbf{v}} + t_{\mathbf{v}\cdot}d}{\alpha_{\mathbf{v}} + c_{\mathbf{v}\cdot}} P_{\sigma(\mathbf{v})}^*(s). \tag{9}$$

## 4 Non-zero Concentration Parameters

In [1] the authors proposed setting all the concentration parameters to zero. Though limiting the flexibility of the model, this allowed them to take advantage of coagulation and fragmentation properties of PYPs [4, 5] to marginalize out all but a linear number (in $T$) of restaurants from the hierarchy. We propose the following enlarged family of hyperparameter settings: let $\alpha_{\varepsilon} = \alpha > 0$ be free to vary at the root of the hierarchy, and set each $\alpha_{\mathbf{u}} = \alpha_{\sigma(\mathbf{u})} d_{\mathbf{u}}$ for each $\mathbf{u} \in \Sigma^* \backslash \{\varepsilon\}$. The

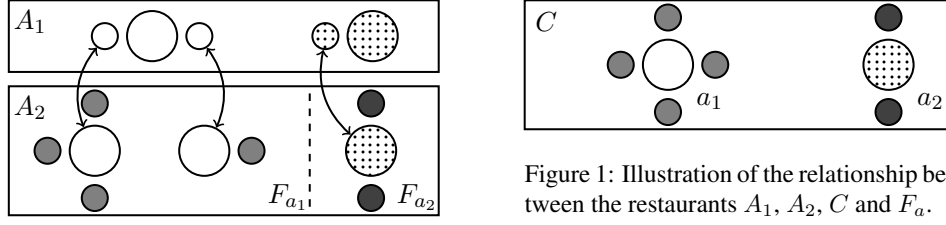

Figure 1: Illustration of the relationship between the restaurants $A_1$, $A_2$, $C$ and $F_a$.

discounts can vary freely. In addition to more flexible modeling, this also partially mitigates the overconfidence problem [2]. To see why, notice from (9) that the predictive probability is a weighted average of predictive probabilities given contexts of various lengths. Since $\alpha_{\mathbf{v}} > 0$, the model gives higher weights to the predictive probabilities of shorter contexts (compared to $\alpha_{\mathbf{v}} = 0$). These typically give less extreme values since they include influences not just from the sequence of identical symbols, but also from other observations of other symbols in other contexts.

Our hyperparameter settings also retain the coagulation and fragmentation properties which allow us to marginalize out many PYPs in the hierarchy for efficient inference. We will provide an elementary proof of these results in terms of CRPs in the following. First we describe the coagulation and fragmentation operations. Let $c \geq 1$ and suppose $A_2 \in \mathcal{A}_c$ and $A_1 \in \mathcal{A}_{|A_2|}$ are two seating arrangements where the number of customers in $A_1$ is the same as that of tables in $A_2$. Each customer in $A_1$ can be put in one-to-one correspondence to a table in $A_2$ and sits at a table in $A_1$. Now consider re-representing $A_1$ and $A_2$. Let $C \in \mathcal{A}_c$ be the seating arrangement obtained by coagulating (merging) tables of $A_2$ corresponding to customers in $A_1$ sitting at the same table. Further, split $A_2$ into sections, one for each table $a \in C$, where each section $F_a \in \mathcal{A}_{|a|}$ contains the $|a|$ customers and tables merged to make up $a$. The converse of coagulating tables of $A_2$ into $C$ is of course to fragment each table $a \in C$ into the smaller tables in $F_a$. Note that there is a one-to-one correspondence between tables in $C$ and in $A_1$, and the number of customers in each table of $A_1$ is that of tables in the corresponding $F_a$. Thus $A_1$ and $A_2$ can be reconstructed from $C$ and $\{F_a\}_{a \in C}$.

**Theorem 1** ([4, 5]). *Suppose $A_2 \in \mathcal{A}_c$, $A_1 \in \mathcal{A}_{|A_2|}$, $C \in \mathcal{A}_c$ and $F_a \in \mathcal{A}_{|a|}$ for each $a \in C$ are related as above. Then the following describe equivalent distributions:*
*(I) $A_2 \sim \mathrm{CRP}_c(\alpha d_2, d_2)$ and $A_1 | A_2 \sim \mathrm{CRP}_{|A_2|}(\alpha, d_1)$.*
*(II) $C \sim \mathrm{CRP}_c(\alpha d_2, d_1 d_2)$ and $F_a | C \sim \mathrm{CRP}_{|a|}(-d_1 d_2, d_2)$ for each $a \in C$.*

*Proof.* We simply show that the joint distributions are the same. Starting with (I) and using (1),

$$
P(A_1, A_2) = \left( \frac{[\alpha + d_1]_{d_1}^{|A_1|-1}}{[\alpha + 1]_1^{|A_2|-1}} \prod_{a \in A_1} [1 - d_1]_1^{|a|-1} \right) \left( \frac{[\alpha d_2 + d_2]_{d_2}^{|A_2|-1}}{[\alpha d_2 + 1]_1^{c-1}} \prod_{b \in A_2} [1 - d_2]_1^{|b|-1} \right)
$$

$$
= \frac{[\alpha d_2 + d_1 d_2]_{d_1 d_2}^{|A_1|-1}}{[\alpha d_2 + 1]_1^{c-1}} \left( \prod_{a \in A_1} [d_2 - d_1 d_2]_{d_2}^{|a|-1} \right) \left( \prod_{b \in A_2} [1 - d_2]_1^{|b|-1} \right).
$$

We used the identity $[\beta \delta + \delta]_\delta^{n-1} = \delta^{n-1} [\beta + 1]_1^{n-1}$ for all $\beta, \delta, n$. Re-grouping the products and expressing the same quantities in terms of $C$ and $\{F_a\}$,

$$
= \frac{[\alpha d_2 + d_1 d_2]_{d_1 d_2}^{|C|-1}}{[\alpha d_2 + 1]_1^{c-1}} \prod_{a \in C} \left( [d_2 - d_1 d_2]_{d_2}^{|F_a|-1} \prod_{b \in F_a} [1 - d_2]_1^{|b|-1} \right) = P(C, \{F_a\}_{a \in C}).
$$

We see that conditioning on $C$ each $F_a \sim \mathrm{CRP}_{|a|}(-d_1 d_2, d_2)$. Marginalizing $\{F_a\}$ out using (1),

$$
P(C) = \frac{[\alpha d_2 + d_1 d_2]_{d_1 d_2}^{|C|-1}}{[\alpha d_2 + 1]_1^{c-1}} \prod_{a \in C} [1 - d_1 d_2]_1^{|a|-1}.
$$

So $C \sim \mathrm{CRP}_c(\alpha d_2, d_1 d_2)$ and *(I)$\Rightarrow$(II)*. Reversing the same argument shows that *(II)$\Rightarrow$(I)*. □

Statement *(I)* of the theorem is exactly the Chinese restaurant franchise of the hierarchical model $G_1 | G_0 \sim \mathrm{PY}(\alpha, d_1, G_0)$, $G_2 | G_1 \sim \mathrm{PY}(\alpha d_2, d_2, G_1)$ with $c$ iid draws from $G_2$. The theorem shows

that the clustering structure of the $c$ customers in the franchise is equivalent to the seating arrangement in a CRP with parameters $\alpha d_2, d_1 d_2$, i.e. $G_2|G_0 \sim \mathrm{PY}(\alpha d_2, d_1 d_2, G_0)$ with $G_1$ marginalized out. Conversely, the fragmentation operation *(II)* regains Chinese restaurant representations for both $G_2|G_1$ and $G_1|G_0$ from one for $G_2|G_0$.

This result can be applied to marginalize out all but a linear number of PYPs from (6) [1]. The resulting model is still a HPYP of the same form as (6), except that it only need be defined over the prefixes of $x_{1:T}$ as well as some subset of their ancestors. In the rest of this paper we will refer to (6) and its Chinese restaurant franchise representation (8) with the understanding that we are operating in this reduced hierarchy. Let $\mathcal{U}$ denote the reduced set of contexts, and redefine $\sigma(\mathbf{u})$ to be the parent of $\mathbf{u}$ in $\mathcal{U}$. The concentration and discount parameters need to be modified accordingly.

## 5    Compact Representation

Current inference algorithms for the SM and hierarchical Pitman-Yor processes operate in the Chinese restaurant franchise representation, and use either Gibbs sampling [3, 11, 1] or particle filtering [2]. To lower memory requirements, instead of storing the precise seating arrangement of each restaurant, the algorithms only store the numbers of customers, numbers of tables and sizes of all tables in the franchise. This is sufficient for sampling and for prediction. However, for large data sets the amount of memory required to store the sizes of the tables can still be very large. We propose algorithms that only store the numbers of customers and tables but not the table sizes. This compact representation needs to store only two integers $(c_{\mathbf{u}s}, t_{\mathbf{u}s})$ per context/symbol pair, as opposed to $t_{\mathbf{u}s}$ integers.[2] These counts are already sufficient for prediction, as (9) does not depend on the table sizes. We will also consider a number of sampling algorithms in this representation.

Our starting point is the joint distribution over the Chinese restaurant franchise (8). Integrating out the seating arrangements $\{A_{\mathbf{u}s}\}$ using (2) gives the joint distribution over $\{c_{\mathbf{u}s}, t_{\mathbf{u}s}\}$:

$$P(\{c_{\mathbf{u}s}, t_{\mathbf{u}s}\}, x_{1:T}) = \left( \prod_{s \in \Sigma} H(s)^{t_{\varepsilon s}} \right) \prod_{\mathbf{u} \in \mathcal{U}} \left( \frac{[\alpha_{\mathbf{u}} + d_{\mathbf{u}}]_{d_{\mathbf{u}}}^{t_{\mathbf{u}\cdot} - 1}}{[\alpha_{\mathbf{u}} + 1]_1^{c_{\mathbf{u}\cdot} - 1}} \prod_{s \in \Sigma} S_{d_{\mathbf{u}}}(c_{\mathbf{u}s}, t_{\mathbf{u}s}) \right). \qquad (10)$$

Note that each $c_{\mathbf{u}s}$ is in fact determined by (7) so in fact the only unobserved variables in (10) are $\{t_{\mathbf{u}s}\}$. With this joint distribution we can now derive various sampling algorithms.

### 5.1    Sampling Algorithms

**Direct Gibbs Sampling of** $\{c_{\mathbf{u}s}, t_{\mathbf{u}s}\}$. It is straightforward derive a Gibbs sampler from (10). Since each $c_{\mathbf{u}s}$ is determined by $c_{\mathbf{u}s}^x$ and the $t_{\mathbf{v}s}$ at child restaurants $\mathbf{v}$, it is sufficient to update each $t_{\mathbf{u}s}$, which for $t_{\mathbf{u}s}$ in the range $\{1, \dots, c_{\mathbf{u}s}\}$ has conditional distribution

$$P(t_{\mathbf{u}s}|\text{rest}) \propto \frac{[\alpha_{\mathbf{u}} + d_{\mathbf{u}}]_{d_{\mathbf{u}}}^{t_{\mathbf{u}\cdot} - 1}}{[\alpha_{\sigma(\mathbf{u})} + 1]_1^{c_{\sigma(\mathbf{u})\cdot} - 1}} S_{d_{\mathbf{u}}}(c_{\mathbf{u}s}, t_{\mathbf{u}s}) S_{d_{\sigma(\mathbf{u})}}(c_{\sigma(\mathbf{u})s}, t_{\sigma(\mathbf{u})s}), \qquad (11)$$

where $t_{\mathbf{u}\cdot}$, $c_{\sigma(\mathbf{u})\cdot}$ and $c_{\sigma(\mathbf{u})s}$ all depend on $t_{\mathbf{u}s}$ through the constraints (7). One problem with this sampler is that we need to compute $S_{d_{\mathbf{u}}}(c, t)$ for all $1 \le c, t \le c_{\mathbf{u}s}$. If $d_{\mathbf{u}}$ is fixed these can be precomputed and stored, but the resulting memory requirement is again large since each restaurant typically has its own $d_{\mathbf{u}}$ value. If $d_{\mathbf{u}}$ is updated in the sampling, then these will need to be computed each time as well, costing $O(c_{\mathbf{u}s}^2)$ per iteration. Further, $S_d(c, t)$ typically has very high dynamic range, so care has to be taken to avoid numerical under-/overflow (e.g. by performing the computations in the log domain, involving many expensive log and exp computations).

**Re-instantiating Seating Arrangements.** Another strategy is to re-instantiate the seating arrangement by sampling $A_{\mathbf{u}s} \sim \mathrm{CRP}_{c_{\mathbf{u}s} t_{\mathbf{u}s}}(d_{\mathbf{u}})$ from its conditional distribution given $c_{\mathbf{u}s}, t_{\mathbf{u}s}$ (see Section 5.2 below), then performing the original Gibbs sampling of seating arrangements [3, 11]. This produces a new number of tables $t_{\mathbf{u}s}$ and the seating arrangement can be discarded. Note however that when $t_{\mathbf{u}s}$ changes this sampler will introduce changes to ancestor restaurants (by adding

or removing customers), so these will need to have their seating arrangements instantiated as well. To implement this sampler efficiently, we visit restaurants in depth-first order, keeping in memory only the seating arrangements of all restaurants on the path to the current one. The computational cost is $O(c_{\mathbf{u}s}t_{\mathbf{u}s})$, but with a potentially smaller hidden constant (no log/exp computations are required).

**Original Gibbs Sampling of $\{c_{\mathbf{u}s}, t_{\mathbf{u}s}\}$.** A third strategy is to "imagine" having a seating arrangement and running the original Gibbs sampler, incrementing $t_{\mathbf{u}s}$ if a table would have been created, and decrementing $t_{\mathbf{u}s}$ if a table would have been deleted. Recall that the original Gibbs sampler operates by iterating over customers, treating each as the last customer in the restaurant, removing it, then adding it back into the restaurant. When removing, if the customer were sitting by himself, a table would need to be deleted too, so the probability of decrementing $t_{\mathbf{u}s}$ is the probability of a customer sitting by himself. From (2), this can be worked out to be

$$P(\text{decrement } t_{\mathbf{u}s}) = \frac{S_{d_{\mathbf{u}}}(c_{\mathbf{u}s} - 1, t_{\mathbf{u}s} - 1)}{S_{d_{\mathbf{u}}}(c_{\mathbf{u}s}, t_{\mathbf{u}s})}. \tag{12}$$

The numerator is due to a sum over all seating arrangements where the other $c_{\mathbf{u}s} - 1$ customers sit at the other $t_{\mathbf{u}s} - 1$ tables. When adding back the customer, the probability of incrementing the number of tables is the probability that the customer sits at a new table of the same dish $s$:

$$P(\text{increment } t_{\mathbf{u}s}) = \frac{(\alpha_{\mathbf{u}} + d_{\mathbf{u}}t_{\mathbf{u}\cdot})P^*_{\sigma(\mathbf{u})}(s)}{(\alpha_{\mathbf{u}} + d_{\mathbf{u}}t_{\mathbf{u}\cdot})P^*_{\sigma(\mathbf{u})}(s) + c_{\mathbf{u}s} - t_{\mathbf{u}s}d_{\mathbf{u}}}, \tag{13}$$

where $P^*_{\sigma(\mathbf{u})}(s)$ is the predictive (9) with the current value of $t_{\mathbf{u}s}$, and $c_{\mathbf{u}s}, t_{\mathbf{u}s}$ are values with the customer removed. This sampler also requires computation of $S_{d_{\mathbf{u}}}(c, t)$, but only for $1 \leq t \leq t_{\mathbf{u}s}$ which can be significantly smaller than $c_{\mathbf{u}s}$. Computation cost is $O(c_{\mathbf{u}s}t_{\mathbf{u}s})$ (but again with a larger constant due to computing the Stirling numbers in a stable way). We did not find a sampling method taking less time than $O(c_{\mathbf{u}s}t_{\mathbf{u}s})$.

**Particle Filtering.** (13) gives the probability of incrementing $t_{\mathbf{u}s}$ (and adding a customer to the parent restaurant) when a customer is added into a restaurant. This can be used as the basis for a particle filter, which iterates through the sequence $x_{1:T}$, adding a customer corresponding to $s = x_i$ in context $\mathbf{u} = x_{1:i-1}$ at each step. Since no customer deletion is required, the cost is very small: just $O(c_{\mathbf{u}s})$ for the $c_{\mathbf{u}s}$ customers per $s$ and $\mathbf{u}$ (plus the cost of traversing the hierarchy to the current restaurant, which is always necessary). Particle filtering works very well in online settings, e.g. compression [2], and as initialization for Gibbs sampling.

## 5.2   Re-instantiating $A_{\mathbf{u}s}$ given $c_{\mathbf{u}s}, t_{\mathbf{u}s}$

To simplify notation, here we will let $d = d_{\mathbf{u}}, c = c_{\mathbf{u}s}, t = t_{\mathbf{u}s}$ and $A = A_{\mathbf{u}s} \in \mathcal{A}_{ct}$. We will use the forward-backward algorithm in an undirected chain to sample $A$ from $\mathrm{CRP}_{ct}(d)$ given in (2). First we re-express $A$ using two sets of variables $z_1, \ldots, z_c$ and $y_1, \ldots, y_c$. Label a table $a \in A$ using the index of the first customer at the table, i.e. the smallest element of $a$. Let $z_i$ be the number of tables occupied by the first $i$ customers, and $y_i$ the label of the table that customer $i$ sits at. The variables satisfy the following constraints: $z_1 = 1$, $z_c = t$, and $z_i = z_{i-1}$ in which case $y_i \in [i-1]$ or $z_i = z_{i-1} + 1$ in which case $y_i = i$. This gives a one-to-one correspondence between seating arrangements in $\mathcal{A}_{ct}$ and settings of the variables satisfying the above constraints. Consider the following distribution over the variables satisfying the constraints: $z_1, \ldots, z_c$ is distributed according to a Markov network with $z_1 = 1$, $z_c = t$, and edge potentials:

$$f(z_i, z_{i-1}) = \begin{cases} i - 1 - z_i d & \text{if } z_i = z_{i-1}, \\ 1 & \text{if } z_i = z_{i-1} + 1, \\ 0 & \text{otherwise.} \end{cases} \tag{14}$$

It is easy to see that the normalization constant is simply $S_d(c, t)$ and

$$P(z_{1:c}) = \frac{\prod_{i:z_i=z_{i-1}}(i - 1 - z_i d)}{S_d(c, t)}. \tag{15}$$

Given $z_{1:c}$, we give each $y_i$ the following distribution conditioned on $y_{1:i-1}$:

$$P(y_i | z_{1:c}, y_{1:i-1}) = \begin{cases} 1 & \text{if } y_i = i \text{ and } z_i = z_{i-1} + 1, \\ \frac{\sum_{j=1}^{i-1} \mathbf{1}(y_j = y_i) - d}{i - 1 - z_i d} & \text{if } z_i = z_{i-1} \text{ and } y_i \in [i-1]. \end{cases} \tag{16}$$

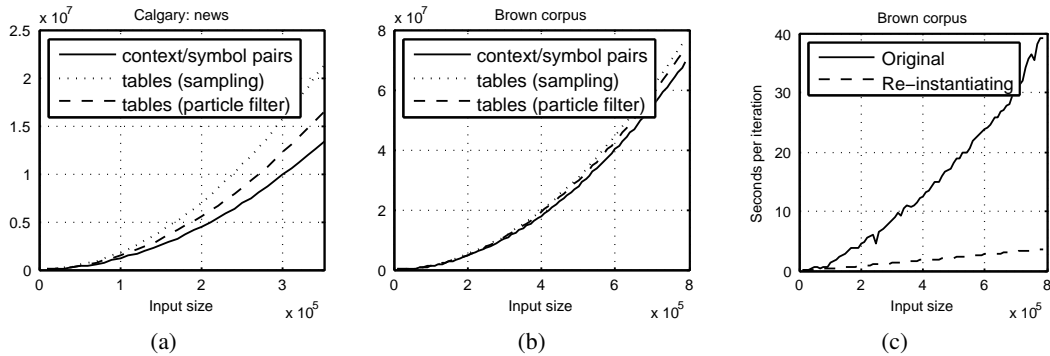

Figure 2: (a), (b) Number of context/symbol pairs and total number of tables (counted after particle filter initialization and 10 sampling iterations using the compact original sampler) as a function input size. Subfigure (a) shows the counts obtained from a byte-level model of the `news` file in the Calgary corpus, whereas (b) shows the counts for word-level model of the Brown corpus (training set). The space required for the compact representation is proportional to the number of context/symbol pairs, whereas for the full representation it is proportional to the number of tables. Note also that sampling tends to increase the number of tables over the particle filter initialization. (c) Time per iteration (seconds) as a function of input size for the original Gibbs sampler in the compact representation and the re-instantiating sampler (on the Brown corpus).

Multiplying all the probabilities together, we see that $P(z_{1:c}, y_{1:c})$ is exactly equal to $P(A)$ in (2). Thus we can sample $A$ by first sampling $z_{1:c}$ from (15), then each $y_i$ conditioned on previous ones using (16), and converting this representation into $A$. We use a backward-filtering-forward-sampling algorithm to sample $z_{1:c}$, as this avoids numerical underflow problems that can arise when using forward-filtering. Backward-filtering avoids these problems by incorporating the constraint that $z_c$ has to equal $t$ into the messages from the beginning.

**Fragmenting a Restaurant.** In particle filtering and in prediction, we often need to re-instantiate a restaurant which was previously marginalized out. We can do so by sampling $A_{\mathbf{u}s}$ given $c_{\mathbf{u}s}, t_{\mathbf{u}s}$ for each $s$, then fragmenting each $A_{\mathbf{u}s}$ using Theorem 1, counting the resulting numbers of customers and tables, then forgetting the seating arrangements.

## 6    Experiments

In order to evaluate the proposed improvements in terms of reduced memory requirements and to compare the performance of the different sampling schemes we performed three sets of experiments.[3] In the first experiment we evaluated the potential space saving due to the compact representation. Figure 2 shows the number of context/symbol pairs and the total number of tables as a function of data set size. While the difference does not seem dramatic, there is still a significant amount of memory that can be saved by using the compact representation, as there is no additional overhead and memory fragmentation due to variable-size arrays. The comparison between the byte-level model and the word-level model in Figure 2 also demonstrates that the compact representation saves more space when $|\Sigma|$ is small (which leads to context/symbol pairs having larger $c_{\mathbf{u}s}$'s and $t_{\mathbf{u}s}$'s). Finally, Figure 2 illustrates another interesting effect: the number of tables is generally larger after a few iterations of Gibbs sampling have been performed after the initialization using a single-particle particle filter [2].

The second experiment compares the computational cost of the compact original sampler and the sampler that re-instantiates full seating arrangements. The main computational cost of the original sampler is computing the ratio (12), while sampling the seating arrangements is the main computational cost of the re-instantiating sampler. Figure 2(c) shows the time needed for one iteration of Gibbs sampling as a function of data set size. The re-instantiating sampler is found to be much more efficient, as it avoids the overhead involved in computing the Stirling numbers in a stable manner (e.g. log/exp computations). For the original sampler, time can be traded off with space

| $\alpha$ | Particle Filter only | | Gibbs (1 sample) | | Gibbs (50 samples averaged) | | Online | |
|---|---|---|---|---|---|---|---|---|
| | Fragment | Parent | Fragment | Parent | Fragment | Parent | PF | Gibbs |
| 0 | 8.45 | 8.41 | 8.44 | 8.41 | 8.43 | 8.39 | 8.04 | 8.04 |
| 1 | 8.41 | 8.39 | 8.40 | 8.39 | 8.39 | 8.38 | 8.01 | 8.01 |
| 3 | 8.37 | 8.37 | 8.37 | 8.37 | 8.35 | 8.35 | 7.98 | 7.98 |
| 10 | 8.33 | 8.34 | 8.33 | 8.33 | 8.32 | 8.32 | 7.95 | 7.94 |
| 20 | 8.32 | 8.33 | 8.32 | 8.32 | 8.31 | 8.31 | 7.94 | 7.94 |
| 50 | 8.32 | 8.33 | 8.31 | 8.32 | 8.31 | 8.31 | 7.95 | 7.95 |

Table 1: Average log-loss on the Brown corpus (test set) for different values of $\alpha$, different inference strategies, and different modes of prediction. Inference is performed by either just using the particle filter or using the particle filter followed by 50 burn-in iterations of Gibbs sampling. Subsequently either 1 or 50 samples are collected for prediction. Prediction is performed either using fragmentation or by predicting from the parent node. The final two columns labelled *Online* show the results obtained by using the particle filter on the test set as well, after training with either just the particle filter or particle filter followed by 50 Gibbs iterations. Non-zero values of $\alpha$ can be seen to provide a significant increase in perfomance, while the gains due to averaging samples or proper fragmentation during prediction are small.

by tabulating all required Stirling numbers along the path down the tree (as was done in these experiments). However, this leads to an additional memory overhead that mostly undoes any savings from the compact representation.

The third set of experiments uses the re-instantiating sampler and compares different modes of prediction and the effect of the non-zero concentration parameter. The results are shown in Table 1. Predictions with the SM can be made in several different ways. After obtaining one or more samples from the posterior distribution over customers and tables (either using particle filtering or Gibbs sampling on the training set) one has a choice of either using particle filtering on the test set as well (online setting), or making predictions while keeping the model fixed. One also has a choice when making predictions involving contexts that were marginalized out from the model: one can either re-instantiate these contexts by fragmentation or simply predict from the parent (or even the child) of the required node. While one ultimately wants to average predictions over the posterior distribution, one may consider using just a single sample for computational reasons.

# 7 Discussion

In this paper we proposed an enlarged set of hyperparameters for the sequence memoizer that retains the coagulation/fragmentation properties important for efficient inference, and we proposed a new minimal representation of the Chinese restaurant processes to reduce the memory requirement of the sequence memoizer. We developed novel inference algorithms for the new representation, and presented experimental results exploring their behaviors. We found that the algorithm which re-instantiates seating arrangements is significantly more efficient than the other two Gibbs samplers, while particle filtering is most efficient but produces slightly worse predictions. Along the way, we formalized the metaphorical language often used to describe Chinese restaurant processes in the machine learning literature, and were able to provide an elementary proof of the coagulation/fragmentation properties. We believe this more precise language will be of use to researchers interested in hierarchical Dirichlet processes and its various generalizations.

We are currently exploring methods to compute or approximate the generalized Stirling numbers, and efficient methods to optimize the hyperparameters in the sequence memoizer. A parting remark is that the posterior distribution over $\{c_{\mathbf{us}}, t_{\mathbf{us}}\}$ in (10) is in the form of a standard Markov network with sum constraints (7). Thus other inference algorithms like loopy belief propagation or variational inference can potentially be applied. There are however two difficulties to be resolved before these are possible: the large domains of the variables, and the large dynamic ranges of the factors.

**Acknowledgments**

We would like to thank the Gatsby Charitable Foundation for generous funding.

## Footnotes

[1]We have omitted the set subscript $\{\cdot\}_{s \in \Sigma}$. We will drop these subscripts when they are clear from context.

[2]In both representations one may also want to store the total number of customers and tables in each restaurant for efficiency. In practice, where there is additional overhead due to the data structures involved, storage space for the full representation can be reduced by treating context/symbol pairs with only one customer separately.

[3]All experiments were performed on two data sets: the `news` file from the Calgary corpus (modeled as a sequence of 377,109 bytes; $|\Sigma| = 256$), and the Brown corpus (preprocessed as in [12]), modeled as a sequence of words (800,000 words training set; 181,041 words test set; $|\Sigma| = 16383$). Following [1], the discount parameters were fixed to .62, .69, .74, .80 for the first 4 levels and .95 for all subsequent levels of the hierarchy.

# References

[1] F. Wood, C. Archambeau, J. Gasthaus, L. F. James, and Y. W. Teh. A stochastic memoizer for sequence data. In *Proceedings of the International Conference on Machine Learning*, volume 26, pages 1129–1136, 2009.

[2] J. Gasthaus, F. Wood, and Y. W. Teh. Lossless compression based on the Sequence Memoizer. In James A. Storer and Michael W. Marcellin, editors, *Data Compression Conference*, pages 337–345, Los Alamitos, CA, USA, 2010. IEEE Computer Society.

[3] Y. W. Teh. A Bayesian interpretation of interpolated Kneser-Ney. Technical Report TRA2/06, School of Computing, National University of Singapore, 2006.

[4] J. Pitman. Coalescents with multiple collisions. *Annals of Probability*, 27:1870–1902, 1999.

[5] M. W. Ho, L. F. James, and J. W. Lau. Coagulation fragmentation laws induced by general coagulations of two-parameter Poisson-Dirichlet processes. http://arxiv.org/abs/math.PR/0601608, 2006.

[6] Y. W. Teh, M. I. Jordan, M. J. Beal, and D. M. Blei. Hierarchical Dirichlet processes. *Journal of the American Statistical Association*, 101(476):1566–1581, 2006.

[7] P. Blunsom, T. Cohn, S. Goldwater, and M. Johnson. A note on the implementation of hierarchical Dirichlet processes. In *Proceedings of the ACL-IJCNLP 2009 Conference Short Papers*, pages 337–340, Suntec, Singapore, August 2009. Association for Computational Linguistics.

[8] J. Pitman and M. Yor. The two-parameter Poisson-Dirichlet distribution derived from a stable subordinator. *Annals of Probability*, 25:855–900, 1997.

[9] H. Ishwaran and L. F. James. Gibbs sampling methods for stick-breaking priors. *Journal of the American Statistical Association*, 96(453):161–173, 2001.

[10] L. C. Hsu and P. J.-S. Shiue. A unified approach to generalized Stirling numbers. *Advances in Applied Mathematics*, 20:366–384, 1998.

[11] Y. W. Teh. A hierarchical Bayesian language model based on Pitman-Yor processes. In *Proceedings of the 21st International Conference on Computational Linguistics and 44th Annual Meeting of the Association for Computational Linguistics*, pages 985–992, 2006.

[12] Y. Bengio, R. Ducharme, P. Vincent, and C. Jauvin. A neural probabilistic language model. *Journal of Machine Learning Research*, 3:1137–1155, 2003.

